# Dopamine Induced Bistability Enhances Signal Processing in Spiny Neurons

**Aaron J. Gruber[1,2], Sara A. Solla[2,3], and James C. Houk[2,1]**
Departments of Biomedical Engineering[1], Physiology[2], and Physics and Astronomy[3]
Northwestern University, Chicago, IL 60201
{ *a-gruber1, solla, j-houk* }*@northwestern.edu*

## Abstract

Single unit activity in the striatum of awake monkeys shows a marked dependence on the expected reward that a behavior will elicit. We present a computational model of spiny neurons, the principal neurons of the striatum, to assess the hypothesis that direct neuromodulatory effects of dopamine through the activation of D1 receptors mediate the reward dependency of spiny neuron activity. Dopamine release results in the amplification of key ion currents, leading to the emergence of bistability, which not only modulates the peak firing rate but also introduces a temporal and state dependence of the model's response, thus improving the detectability of temporally correlated inputs.

## 1 Introduction

The classic notion of the basal ganglia as being involved in purely motor processing has expanded over the years to include sensory and cognitive functions. A surprising new finding is that much of this activity shows a motivational component. For instance, striatal activity related to visual stimuli is dependent on the type of reinforcement (primary vs secondary) that a behavior will elicit [1]. Task-related activity can be enhanced or suppressed when a reward is anticipated for correct performance, relative to activity when no reward is expected. Although the origin of this reward dependence has not been experimentally verified, dopamine modulation is likely to play a role. Spiny neurons in the striatum, the input to the basal ganglia, receive a prominent neuromodulatory input from dopamine neurons in the substantia nigra pars compacta. These dopamine neurons discharge in a reward-dependent manner [2]; they respond to the delivery of unexpected rewards and to sensory cues that reliably precede the delivery of expected rewards.

Activation of dopamine receptors alters the response characteristics of spiny neurons by modulating the properties of voltage-gated ion channels, as opposed to simple excitatory or inhibitory effects [3]. Activation of the D1 type dopamine receptor alone can either enhance or suppress neural responses depending on the prior state of the spiny neuron [4]. Here, we use a computational approach to assess the hypothesis that the modulation of specific ion channels through the activation of D1 receptors is sufficient to explain both the enhanced and suppressed single unit

responses of medium spiny neurons to reward-predicting stimuli.

We have constructed a biophysically grounded model of a spiny neuron and used it to investigate whether dopamine neuromodulation accounts for the observed reward-dependence of striatal single-unit responses to visual targets in the memory guided saccade task described by [1]. These authors used an asymmetric reward schedule and compared the response to a given target in rewarded as opposed to unrewarded cases. They report a substantial reward-dependent difference; the majority of these neurons showed a reward-related enhancement of the intensity and duration of discharge, and a smaller number exhibited a reward-related depression. The authors speculated that D1 receptor activation might account for enhanced responses, whereas D2 receptor activation might explain the depressed responses. The model presented here demonstrates that neuromodulatory actions of dopamine through D1 receptors suffice to account for both effects, with interesting consequences for information processing.

## 2 Model description

The membrane properties of the model neuron result from an accurate representation of a minimal set of currents needed to reproduce the characteristic behavior of spiny neurons. In low dopamine conditions, these cells exhibit quasi two-state behavior; they spend most of their time either in a hyperpolarized 'down' state around $-85\,mV$, or in a depolarized 'up' state around $-55\,mV$ [5]. This bimodal character of the response to cortical input is attributed to a combination of inward rectifying (IRK) and outward rectifying (ORK) potassium currents [5]. IRK contributes a small outward current at hyperpolarized membrane potentials, thus providing resistance to depolarization and stabilizing the down state. ORK is a major hyperpolarizing current that becomes activated at depolarized potentials and opposes the depolarizing influences of excitatory synaptic and inward ionic currents; it is their balance that determines the membrane potential of the up state. In addition to IRK and ORK currents, the model incorporates the L-type calcium (L-Ca) current that starts to provide an inward current at subthreshold membrane potentials, thus determining the voltage range of the up state. This current has the ability to increase the firing rate of spiny neurons and is critical to the enhancement of spiny neuron responses in the presence of D1 agonists [4].

Our goal is to design a model that provides a consistent description of membrane properties in the 100 - 1000 $ms$ time range. This is the characteristic range of duration for up and down state episodes; it also spans the time course of short term modulatory effects of dopamine. The model is constructed according to the principle of separation of time scales: processes that operate in the 100-1000 $ms$ range are modeled as accurately as possible, those that vary on a much shorter time scale are assumed to instantaneously achieve their steady-state values, and those that occur over longer time scales, such as slow inactivation, are assumed constant. Thus, the model does not incorporate currents which inactivate on a short time scale, and cannot provide a good description of rapid events such as the transitions between up and down states or the generation of action potentials.

The membrane of a spiny neuron is modeled here as a single compartment with steady-state voltage-gated ion currents. A first order differential equation relates the temporal change in membrane potential ($V_m$) to the membrane currents ($I_i$),

$$-C_m \frac{dV_m}{dt} = \gamma \left(I_{IRK} + I_{L-Ca}\right) + I_{ORK} + I_l + I_s.$$

(1)

The right hand side of the equation includes active ionic, leakage, and synaptic currents. The multiplicative factor $\gamma$ models the modulatory effects of D1 receptor activation by dopamine, to be described in more detail later. Ionic currents are modeled using a standard formulation; the parameters are as reported in the biophysical literature, except for adjustments that compensate for specific experimental conditions so as to more closely match in vivo realizations.

All currents except for L-Ca are modeled by the product of a voltage gated conductance and a linear driving force, $I_i = g_i (V_m - E_i)$, where $E_i$ is the reversal potential of ion species $i$ and $g_i$ is the corresponding conductance. The leakage conductance is constant; the conductances for IRK and ORK are voltage gated, $g_i = \bar{g}_i \mathcal{L}_i (V_m)$, where $\bar{g}_i$ is the maximum conductance and $\mathcal{L}_i (V_m)$ is a logistic function of the membrane potential. Calcium currents are not well represented by a linear driving force model; extremely low intracellular calcium concentrations result in a nonlinear driving force well accounted for by the Goldman-Hodgkin-Katz equation [6],

$$I_{L-Ca} = \bar{P}_{L-Ca} \mathcal{L}_{L-Ca} (V_m) \left( \frac{z^2 V_m F^2}{RT} \right) \left( \frac{[Ca]_i - [Ca]_o e^{-\frac{z V_m F}{RT}}}{1 - e^{-\frac{z V_m F}{RT}}} \right), \qquad (2)$$

where $\bar{P}_{L-Ca}$ is the maximum permeability. The resulting ionic currents are shown in Fig 1A.

The synaptic current is modeled as the product of a conductance and a linear driving force, $I_s = g_s(V_m - E_s)$, with $E_s = 0$. The synaptic conductance includes two types of cortical input: a phasic sensory-related component $g_p$, and a tonic context-related component $g_t$, which are added to determine the total synaptic input: $g_s = \xi(g_p + g_t)$. The factor $\xi$ is a random variable that simulates the noisy character of synaptic input.

Dopamine modulates the properties of ion currents though the activation of specific receptors. Agonists for the D1 type receptor enhance the IRK and L-Ca currents observed in spiny neurons [7, 8]. This effect is modeled by the factor $\gamma$ in Eq 1. An upper bound of $\gamma = 1.4$ is derived from physiological experiments [7, 8]. The lower bound at $\gamma = 1.0$ corresponds to low dopamine levels; this is the experimental condition in which the ion currents have been characterized.

## 3   Static and dynamic properties

Stationary solutions to Eq 1 correspond to equilibrium values of the membrane potential $V_m$ consistent with specific values of the dopamine controlled conductance gain parameter $\gamma$ and the total synaptic conductance $g_s$; fluctuations of $g_s$ around its mean value are ignored in this section: the noise parameter is set to $\xi = 1$. Stationary solutions satisfy $dV_m/dt = 0$; it follows from Eq 1 that they result from

$$\gamma(I_{IRK} + I_{L-Ca}) + I_{ORK} + I_l = -g_s V_m. \qquad (3)$$

Intersections between a curve representing the total ionic current (left hand side of Eq 3) as a function of $V_m$ and a straight line representing the negative of the synaptic current (right hand side of Eq 3) determine the stationary values of the membrane potential. Solutions to Eq 3 can be followed as a function of $g_s$ for fixed $\gamma$ by varying the slope of the straight line. For $\gamma = 1$ there is only one such intersection for any value of $g_s$. At low dopamine levels, $V_m$ is a single-valued monotonically increasing function of $g_s$, shown in Fig 1B (dotted line). This operational curve describes a

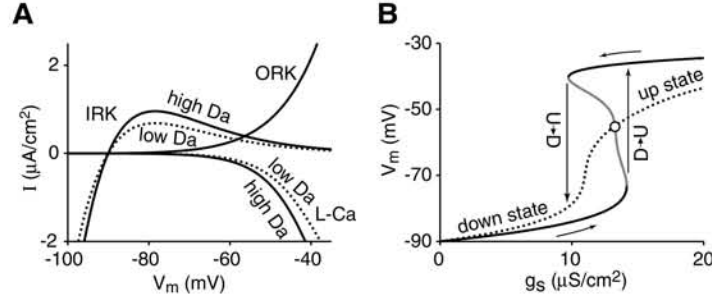

Figure 1: Model characterization in low ($\gamma = 1.0$, dotted lines) and high ($\gamma = 1.4$, solid lines) dopamine conditions. (A) Voltage-gated ion currents. (B) Operational curves: stationary solutions to Eq 1.

gradual, smooth transition from hyperpolarized values of $V_m$ corresponding to the down state to depolarized values of $V_m$ corresponding to the up state. At high dopamine levels ($\gamma = 1.4$), the membrane potential is a single-valued monotonically increasing function of the synaptic conductance for either $g_s < 9.74 \ \mu S/cm^2$ or $gs > 14.17 \ \mu S/cm^2$. In the intermediate regime $9.74 \ \mu S/cm^2 < g_s < 14.17 \ \mu S/cm^2$, there are three solutions to Eq 3 for each value of $g_s$. The resulting operational curve, shown Fig 1B (solid line), consists of three branches: two stable and one unstable. The two stable branches (dark solid lines) correspond to a hyperpolarized down state (lower branch) and a depolarized up state (upper branch). The unstable branch (solid gray line) corresponds to intermediate values of $V_m$ that are not spontaneously sustainable.

Bistability arises through a saddle node bifurcation with increasing $\gamma$ and has a drastic effect on the response properties of the model neuron in high dopamine conditions. Consider an experiment in which $\gamma$ is fixed at 1.4 and $g_s$ changes slowly so as to allow $V_m$ to follow its equilibrium value on the operational curve for $\gamma = 1.4$ (see Fig 1B). As $g_s$ increases, the hyperpolarized down state follows the lower stable branch. As $g_s$ reaches $14.17 \ \mu S/cm^2$, the synaptic current suddenly overcomes the mostly IRK hyperpolarizing current, and $V_m$ depolarizes abruptly to reach an up state stabilized by the activation of the hyperpolarizing ORK current. This is the down to up (D→U) state transition. As $g_s$ is increased further, the up state follows the upper stable branch, with a small amount of additional depolarization. If $g_s$ is now decreased, the depolarized up state follows the stable upper branch in the downward direction. It is the inward L-Ca current which counteracts the hyperpolarizing effect of the ORK current and stabilizes the up state until $g_s$ reaches $9.74 \ \mu S/cm^2$, where a net hyperpolarizing ionic current overtakes the system and $V_m$ hyperpolarizes abruptly to the down state. This is the up to down (U→D) state transition.

The emergence of bistability in high dopamine conditions results in a prominent hysteresis effect. The state of the model, as described by the value of $V_m$, depends not only on the current values of $\gamma$ and $g_s$, but also on the particular trajectory followed by these parameters to reach their current values. The appearance of bistability gives a well defined meaning to the notion of a down state and an up state: in this case there is a gap between the two stable branches, while in low dopamine conditions the transition is smooth, with no clear separation between states. We generically refer to hyperpolarized potentials as the down state and depolarized potentials as the up state, for consistency with the electrophysiological terminology.

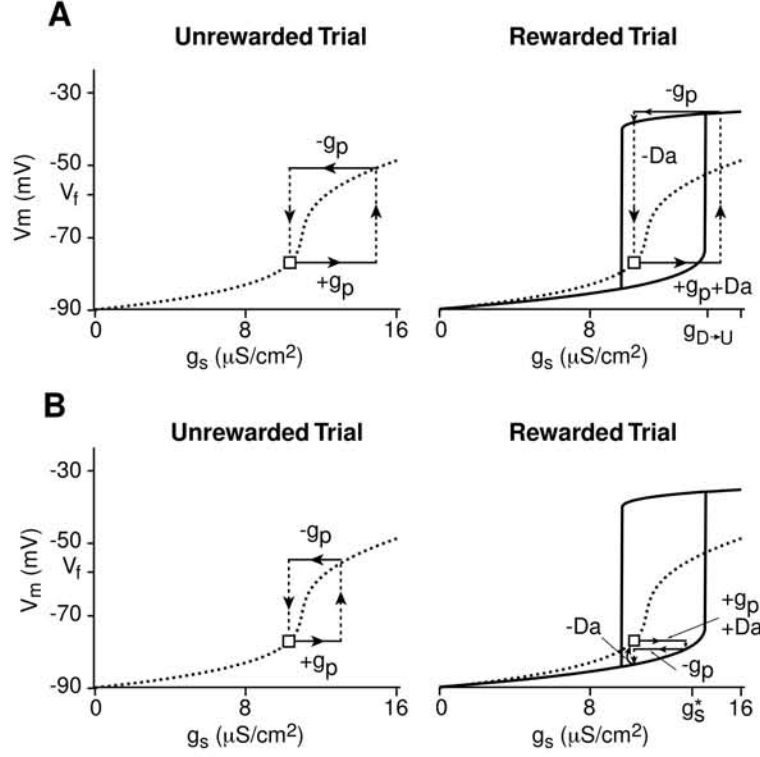

Figure 2: Response to a sensory related phasic input in rewarded and unrewarded trials. (A) $g_t + g_p > g_{D \to U}$. (B) $g_t + g_p < g_s^*$.

An important feature of the model is that operational curves for all values of $\gamma$ intersect at a unique point, indicated by a circle in Fig 1B, for which $V_m^* = -55.1 \ mV$ and $g_s^* = 13.2 \ \mu S/cm^2$. The appearance of this *critical point* is due to a perfect cancellation between the IRK and the L-Ca currents; it arises as a solution to the equation $I_{IRK} + I_{L-Ca} = 0$. When this condition is satisfied, solutions to Eq 3 become independent of $\gamma$. The existence of a critical point at a slightly more depolarized membrane potential than the firing threshold at $V_f = -58 \ mV$ is an important aspect of our model; it plays a role in the mechanism that allows dopamine to either enhance or depress the response of the model spiny neuron.

The dynamical evolution of $V_m$ due to changes in both $g_s$ and $\gamma$ follows from Eq 1. Consider a scenario in which a tonic input $g_t$ maintains $V_m$ below $V_f$; the response to an additional phasic input $g_p$ sufficient to drive $V_m$ above $V_f$ depends on whether it is associated with expected reward and thus triggers dopamine release. The response of the model neuron depends on the combined synaptic input $g_s$ in a manner that is critically dependent on the expectation of reward. We consider two cases: whether $g_s$ exceeds $g_{D \to U}$ (Fig 2A) or remains below $g_s^*$ (Fig 2B).

If the phasic input is not associated with reward, the dopamine level does not increase (left panels in Fig 2). The square on the operational curve for $\gamma = 1$ (dotted line) indicates the equilibrium state corresponding to $g_t$. A rapid increase from $g_s = g_t$ to $g_s = g_t + g_p$ (rightward solid arrow) is followed by an increase in $V_m$ towards its equilibrium value (upward dotted arrow). When the phasic input is removed (leftward solid arrow), $V_m$ decreases to its initial equilibrium value (down-

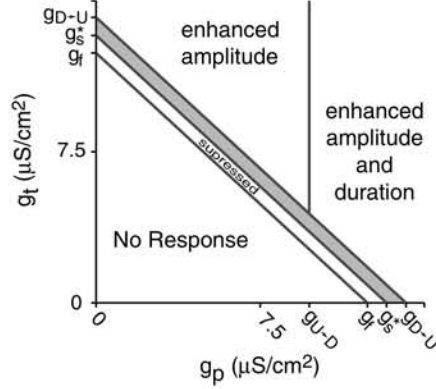

Figure 3: Modulation of response in high dopamine relative to low dopamine conditions as a function of the strength of phasic and tonic inputs.

ward dotted arrow). In unrewarded trials, the only difference between a larger and a smaller phasic input is that the former results in a more depolarized membrane potential and thus a higher firing rate. The firing activity, which ceases when the phasic input disappears, encodes for the strength of the sensory-related stimulus.

Rewarded trials (right panels in Fig 2) elicit qualitatively different responses. The phasic input is the conditioned stimulus that triggers dopamine release in the striatum, and the operational curve switches from the $\gamma = 1$ (dotted) curve to the bistable $\gamma = 1.4$ (solid) curve. The consequences of this switch depend on the strength of the phasic input. If $g_s$ exceeds the value for the D→U transition (Fig 2A), $V_m$ depolarizes towards the upper branch of the bistable operational curve. This additional depolarization results in a noticeably higher firing rate than the one elicited by the same input in an unrewarded trial (Fig 2A, left panel). When the phasic input is removed, the unit hyperpolarizes slightly as it reaches the upper branch of the bistable operational curve. If $g_t$ exceeds $g_{U \to D}$, the unit remains in the up state until $\gamma$ decreases towards its baseline level. If this condition is met in a rewarded trial, the response is not only larger in amplitude but also longer in duration. In contrast to these enhancements, if $g_s$ is not sufficient to exceed $g_s^*$ (Fig 2B), $V_m$ hyperpolarizes towards the lower branch of the bistable operational curve. The unit remains in the down state until $\gamma$ decreases towards its baseline level. In this type of rewarded trial, dopamine suppresses the response of the unit.

The analysis presented above provides an explanatory mechanism for the observation of either enhanced or suppressed spiny neuron activity in the presence of dopamine. It is the strength of the total synaptic input that selects between these two effects; the generic features of their differentiation are summarized in Fig 3. Enhancement occurs whenever the condition $g_s > g_{D \to U}$ is met, while activity is suppressed if $g_s < g_s^*$. The separatrix between enhancement and suppression always lies in a narrow band limited by $g_s = g_{D \to U}$ and $g_s = g_s^*$. Its precise location will depend on the details of the temporal evolution of $\gamma$ as it rises and returns to baseline. But whatever the shape of $\gamma(t)$ might be, there will be a range of values of $g_s$ for which activity is suppressed, and a range of values of $g_s$ for which activity is enhanced.

# 4    Information processing

Dopamine induced bistability improves the ability of the model spiny neuron to detect time correlated sensory-related inputs relative to a context-related background. To illustrate this effect, consider $g_s = \xi(g_t + g_p)$ as a random variable. The multiplicative noise is Gaussian, with $<\xi> = 1$ and $<\xi^2> = 1.038$. The total probability

density function (PDF) shown in Fig 4A for $g_t = 9.2\ \mu S/cm^2$ consists of two PDFs corresponding to $g_p = 0$ (left; black line) and $g_p = 5.8\ \mu S/cm^2$ (right; grey line). These two values of $g_p$ occur with equal prior probability; time correlations are introduced through a repeat probability $p_r$ of retaining the current value of $g_p$ in the subsequent time step. The total PDF shown in Fig 4A does not depend on the value of $p_r$. Performance at the task of detecting the sensory-related input ($g_p \neq 0$) is limited by the overlap of the corresponding PDFs [9]; optimal separation of the two PDFs in Fig 4A results in a Bayesian error of 10.46%.

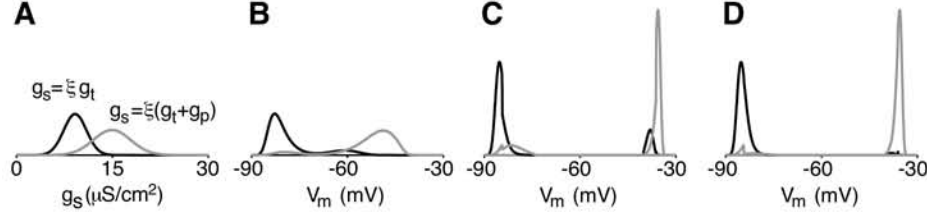

Figure 4: Probability density functions for (A) synaptic input, (B) membrane potential at $\gamma = 1$, (C) membrane potential at $\gamma = 1.4$ for uncorrelated inputs ($p_r = 0.5$), and (D) membrane potential at $\gamma = 1.4$ for correlated inputs ($p_r = 0.975$).

The transformation of $g_s$ into $V_m$ through the $\gamma = 1$ operational curve results in the PDFs shown in Fig 4B; here again, the total PDF does not depend on $p_r$. An increase in the separation of the two peaks indicates an improved signal-to-noise ratio, but an extension in the tails of the PDFs counteracts this effect: the Bayesian error stays at 10.46%, in agreement with theoretical predictions [9] that hold for any strictly monotonic map from $g_s$ into $V_m$. For the $\gamma = 1.4$ operational curve, the PDFs that characterize $V_m$ depend on $p_r$ and are shown in Fig 4C ($p_r = 0.5$, for which $g_p$ is independently drawn from its prior in each time step) and 4D ($p_r = 0.975$, which describes phasic input persistance for about 400 $ms$). The implementation of Bayesian optimal detection of $g_p \neq 0$ for $\gamma = 1.4$ requires three separating boundaries; the corresponding Bayesian errors stand at 10.46% for Fig 4C and 4.23% for Fig 4D. A single separating boundary in the gap between the two stable branches is suboptimal, but is easily implementable by the bistable neuron. This strategy leads to detection errors of 20.06% for Fig 4C and 4.38% for Fig 4D. Note that the Bayesian error decreases only when time correlations are included, and that in this case, detection based on a single separating boundary is very close to optimal. The results for $\gamma = 1.4$ clearly indicate that ambiguities in the bistable region make it harder to identify temporally uncorrelated instances of $g_p \neq 0$ on the basis of a single separating boundary (Fig 4C), while performance improves if instances with $g_p \neq 0$ are correlated over time (Fig 4D). Bistability thus provides a mechanism for improved detection of time correlated input signals.

## 5 Conclusions

The model presented here incorporates the most relevant effects of dopamine neuromodulation of striatal medium spiny neurons via D1 receptor activation. In the absence of dopamine the model reproduces the bimodal character of medium spiny neurons [5]. In the presence of dopamine, the model undergoes a bifurcation and becomes bistable. This qualitative change in character provides a mechanism to account for both enhancement and depression of spiny neuron discharge in response to inputs associated with expectation of reward. There is only limited direct experimental evidence of bistability in the membrane potential of spiny neurons: the

sustained depolarization observed in vitro following brief current injection in the presence of D1 agonists [4] is a hallmark of bistable responsiveness.

The activity of single striatal spiny neurons recorded in a memory guided saccade task [1] is strongly modulated by the expectation of reward as reinforcement for correct performance. In these experiments, most units show a more intense response of longer duration to the presentation of visual stimuli indicative of upcoming reward; a few units show instead suppressed activity. These observations are consistent with properties of the model neuron, which is capable of both types of response to such stimuli. The model identifies the strength of the total excitatory cortical input as the experimental parameter that selects between these two response types, and suggests that enhanced responses can have a range of amplitudes but attenuated responses result in an almost complete suppression of activity, in agreement with experimental data [1].

Bistability provides a gain mechanism that nonlinearly amplifies both the intensity and duration of striatal activity. This amplification, exported through thalamocortical pathways, may provide a mechanism for the preferential cortical encoding of salient information related to reward acquisition. The model indicates that through the activation of D1 receptors, dopamine can temporarily desensitize spiny neurons to weak inputs while simultaneously sensitizing spiny neurons to large inputs. A computational advantage of this mechanism is the potential adaptability of signal modulation: the brain may be able to utilize the demonstrated plasticity of corticostriatal synapses so that dopamine release preferentially enhances salient signals related to reward. This selective enhancement of striatal activity would result in a more informative efferent signal related to achieving reward.

At the systems level, dopamine plays a significant role in the normal operation of the brain, as evident in the severe cognitive and motor deficits associated with pathologies of the dopamine system (e.g. Parkinson's disease, schizophrenia). Yet at the cellular level, the effect of dopamine on the physiology of neurons seems modest. In our model, a small increase in the magnitude of both IRK and L-Ca currents elicited by D1 receptor activation suffices to switch the character of spiny neurons from bimodal to truly bistable, which not only modulates the frequency of neural responses but also introduces a state dependence and a temporal effect. Other models have suggested that dopamine modulates contrast [9], but the temporal effect is a novel aspect that plays an important role in information processing.

# 6  References

[1] Kawagoe R, Takikawa Y, Hikosaka O (1998). *Nature Neurosci* 1:411-416.

[2] Schultz W (1998). *J Neurophysiol* 80:1-27.

[3] Nicola SM, Surmeier DJ, Malenka RC (2000). *Annu Rev Neurosci* 23:185-215.

[4] Hernández-López S, Bargas J, Surmeier DJ, Reyes A, Gallarraga E (1997). *J Neurosci* 17:3334-42.

[5] Wilson CJ, Kawaguchi Y (1996). *J Neurosci* 7:2397-2410.

[6] Hille B (1992) *Ionic Channels of Excitable Membranes*. Sinauer Ass., Sunderland MA.

[7] Pacheco-Cano MT, Bargas J, Hernández-López S (1996). *Exp Brain Res* 110:205-211.

[8] Surmeier DJ, Bargas J, Hemmings HC, Nairn AC, Greengard P (1995). *Neuron* 14:385-397.

[9] Servan-Schreiber D, Printz H, Cohen JD (1990). *Science* 249:892-895.
